# Learning on Graph with Laplacian Regularization

**Rie Kubota Ando**
IBM T.J. Watson Research Center
Hawthorne, NY 10532, U.S.A.
rie1@us.ibm.com

**Tong Zhang**
Yahoo! Inc.
New York City, NY 10011, U.S.A.
tzhang@yahoo-inc.com

## Abstract

We consider a general form of transductive learning on graphs with Laplacian regularization, and derive margin-based generalization bounds using appropriate geometric properties of the graph. We use this analysis to obtain a better understanding of the role of normalization of the graph Laplacian matrix as well as the effect of dimension reduction. The results suggest a limitation of the standard degree-based normalization. We propose a remedy from our analysis and demonstrate empirically that the remedy leads to improved classification performance.

## 1  Introduction

In graph-based methods, one often constructs *similarity graphs* by linking similar data points that are close in the feature space. It was proposed in [3] that one may first project these data points into the eigenspace corresponding to the largest eigenvalues of a normalized adjacency matrix of the graph and then use the standard $k$-means method for clustering. In the ideal case, points in the same class will be mapped into a single point in the reduced eigenspace, while points in different classes will be mapped to different points. One may also consider similar ideas in semi-supervised learning using a discriminative kernel method. If the underlying kernel is induced from the graph, one may formulate semi-supervised learning directly on the graph (e.g., [1, 5, 7, 8]). In these studies, the kernel is induced from the adjacency matrix $\mathbf{W}$ whose $(i, j)$-entry is the weight of edge $(i, j)$. $\mathbf{W}$ is sometimes normalized by $\mathbf{D}^{-1/2}\mathbf{W}\mathbf{D}^{-1/2}$ [2, 4, 3, 7] where $\mathbf{D}$ is a diagonal matrix whose $(j, j)$-entry is the degree of the $j$-th node, but sometimes not [1, 8]. Although such normalization may significantly affect the performance, the issue has not been studied from the learning theory perspective. The relationship of kernel design and graph learning was investigated in [6], which argued that quadratic regularization-based graph learning can be regarded as kernel design. However, normalization of $\mathbf{W}$ was not considered there. The goal of this paper is to provide some learning theoretical insight into the role of normalization of the graph Laplacian matrix $(\mathbf{D} - \mathbf{W})$. We first present a model for transductive learning on graphs and develop a margin analysis for multi-class graph learning. Based on this, we analyze the performance of Laplacian regularization-based graph learning in relation to graph properties. We use this analysis to obtain a better understanding of the role of normalization of the graph Laplacian matrix as well as dimension reduction in graph learning. The results indicate a limitation of the commonly practiced degree-based normalization mentioned above. We propose a learning theoretical remedy based on our analysis and use experiments to demonstrate that the remedy leads to improved classification performance.

## 2  Transductive Learning Model

We consider the following multi-category transductive learning model defined on a graph. Let $V = \{v_1, \ldots, v_m\}$ be a set of $m$ nodes, and let $\mathcal{Y}$ be a set of $K$ possible output values. Assume that each node $v_j$ is associated with an output value $y_j \in \mathcal{Y}$, which we are interested in predicting. We randomly draw a set of $n$ indices $Z_n = \{j_i : 1 \le i \le n\}$ from $\{1, \ldots, m\}$ uniformly and without

replacement. We then manually label the $n$ nodes $v_{j_i}$ with labels $y_{j_i} \in \mathcal{Y}$, and then automatically label the remaining $m - n$ nodes. The goal is to estimate the labels on the remaining $m - n$ nodes as accurately as possible. We encode the label $y_j$ into a vector in $R^K$, so that the problem becomes that of generating an estimation vector $f_{j,\cdot} = [f_{j,1}, \ldots, f_{j,K}] \in R^K$, which can then be used to recover the label $y_j$. In multi-category classification with $K$ classes $\mathcal{Y} = \{1, \ldots, K\}$, we encode each $y_j = k \in \mathcal{Y}$ as $e_k \in R^K$, where $e_k$ is a vector of zero entries except for the $k$-th entry being one. Given $f_{j,\cdot} = [f_{j,1}, \ldots, f_{j,K}] \in R^K$ (which is intended to approximate $e_{y_j}$), we decode the corresponding label estimation $\hat{y}_j$ as: $\hat{y}_j = \arg\max_k \{f_{j,k} : k = 1, \ldots, K\}$. If the true label is $y_j$, then the classification error is $\mathbf{err}(f_{j,\cdot}, y_j) = I(\hat{y}_j \neq y_j)$, where we use $I(\cdot)$ to denote the set indicator function.

In order to estimate $f = [f_{j,k}] \in R^{mK}$ from only a subset of labeled nodes, we consider for a given kernel matrix $\mathbf{K} \in R^m$, the quadratic regularization $f^T \mathbf{Q_K} f = \sum_{k=1}^{K} f_{\cdot,k}^T \mathbf{K}^{-1} f_{\cdot,k}$, where $f_{\cdot,k} = [f_{1,k}, \ldots, f_{m,k}] \in R^m$. We assume that $\mathbf{K}$ is full-rank. We will consider the kernel matrix induced by the graph Laplacian, to be introduced later in the paper. Note that the bold symbol $\mathbf{K}$ denotes the kernel matrix, and regular $K$ denotes the number of classes.

Given a vector $f \in R^{mK}$, the accuracy of its component $f_{j,\cdot} = [f_{j,1}, \ldots, f_{j,K}] \in R^K$ is measured by a loss function $\phi(f_{j,\cdot}, y_j)$. Our learning method attempts to minimize the empirical risk on the set $Z_n$ of $n$ labeled training nodes, subject to $f^T \mathbf{Q_K} f$ being small:

$$\hat{f}(Z_n) = \arg\min_{f \in R^{mK}} \left[ \frac{1}{n} \sum_{j \in Z_n} \phi(f_{j,\cdot}, y_j) + \lambda f^T \mathbf{Q_K} f \right]. \tag{1}$$

where $\lambda > 0$ is an appropriately chosen regularization parameter.

In this paper, we focus on a special class of loss function that is of the form $\phi(f_{j,\cdot}, y_j) = \sum_{k=1}^{K} \phi_0(f_{j,k}, \delta_{k,y_j})$, where $\delta_{a,b}$ is the delta function defined as: $\delta_{a,b} = 1$ when $a = b$ and $\delta_{a,b} = 0$ otherwise. We are interested in the generalization behavior of (1) compared to a properly defined optimal regularized risk, often referred to as "oracle inequalities" in the learning theory literature.

**Theorem 1** *Let $\phi(f_{j,\cdot}, y_j) = \sum_{k=1}^{K} \phi_0(f_{j,k}, \delta_{k,y_j})$ in (1). Assume that there exist positive constants $a$, $b$, and $c$ such that: (i) $\phi_0(x, y)$ is non-negative and convex in $x$, (ii) $\phi_0(x, y)$ is Lipschitz with constant $b$ when $\phi_0(x, y) \leq a$, and (iii) $c = \inf\{x : \phi_0(x, 1) \leq a\} - \sup\{x : \phi_0(x, 0) \leq a\}$. Then $\forall p > 0$, the expected generalization error of the learning method (1) over the random training samples $Z_n$ can be bounded by:*

$$\mathbf{E}_{Z_n} \frac{1}{m-n} \sum_{j \in \bar{Z}_n} \mathbf{err}(\hat{f}_{j,\cdot}(Z_n), y_j) \leq \frac{1}{a} \inf_{f \in R^{mK}} \left[ \frac{1}{m} \sum_{j=1}^{m} \phi_0(f_{j,\cdot}, y_j) + \lambda f^T \mathbf{Q_K} f \right] + \left( \frac{b \mathbf{tr}_p(\mathbf{K})}{\lambda n c} \right)^p,$$

*where $\bar{Z}_n = \{1, \ldots, m\} - Z_n$, $\mathbf{tr}_p(\mathbf{K}) = \left( \frac{1}{m} \sum_{j=1}^{m} \mathbf{K}_{j,j}^p \right)^{1/p}$, and $\mathbf{K}_{j,j}$ is the $(j, j)$-entry of $\mathbf{K}$.*

*Proof.* The proof is similar to the proof of a related bound for binary-classification in [6]. We shall introduce the following notation. let $i_{n+1} \neq i_1, \ldots, i_n$ be an integer randomly drawn from $\bar{Z}_n$, and let $Z_{n+1} = Z_n \cup \{i_{n+1}\}$. Let $\hat{f}(Z_{n+1})$ be the semi-supervised learning method (1) using training data in $Z_{n+1}$: $\hat{f}(Z_{n+1}) = \arg\inf_{f \in R^{mK}} \left[ \frac{1}{n} \sum_{j \in Z_{n+1}} \phi(f_{j,\cdot}, Y_j) + \lambda f^T \mathbf{Q_K} f \right]$. Adapted from a related lemma used in [6] for proving a similar result, we have the following inequality for each $k = 1, \ldots, K$:

$$|\hat{f}_{i_{n+1},k}(Z_{n+1}) - \hat{f}_{i_{n+1},k}(Z_n)| \leq |\nabla_{1,k} \phi(\hat{f}_{i_{n+1},\cdot}(Z_{n+1}), Y_{i_{n+1}})| \mathbf{K}_{i_{n+1},i_{n+1}}/(2\lambda n), \tag{2}$$

where $\nabla_{1,k} \phi(f_{i,\cdot}, y)$ denotes a sub-gradient of $\phi(f_{i,\cdot}, y)$ with respect to $f_{i,k}$, where $f_{i,\cdot} = [f_{i,1}, \ldots, f_{i,K}]$. Next we prove

$$\mathbf{err}(\hat{f}_{i_{n+1},\cdot}(Z_n), y_{i_{n+1}}) \leq \sup_{k = k_0, i_{n+1}} \left[ \frac{1}{a} \phi_0(\hat{f}_{i_{n+1},k}(Z_{n+1}), \delta_{i_{n+1},k}) + \left( \frac{b}{c\lambda n} \mathbf{K}_{i_{n+1},i_{n+1}} \right)^p \right]. \tag{3}$$

In fact, if $\hat{f}(Z_n)$ does not make an error on the $i_{n+1}$-th example, then the inequality automatically holds. Otherwise, assume that $\hat{f}(Z_n)$ makes an error on the $i_{n+1}$-th example, then there exists $k_0 \neq$

$y_{i_{n+1}}$ such that $\hat{f}_{i_{n+1},y_{i_{n+1}}}(Z_n) \leq \hat{f}_{i_{n+1},k_0}(Z_n)$. If we let $d = (\inf\{x : \phi_0(x,1) \leq a\} + \sup\{x : \phi_0(x,0) \leq a\})/2$, then either $\hat{f}_{i_{n+1},y_{i_{n+1}}}(Z_n) \leq d$ or $\hat{f}_{i_{n+1},k_0}(Z_n) \geq d$. By the definition of $c$ and $d$, it follows that there exists $k = k_0$ or $k = i_{n+1}$ such that either $\phi_0(\hat{f}_{i_{n+1},k}(Z_{n+1}), \delta_{i_{n+1},k}) \geq a$ or $\left| \hat{f}_{i_{n+1},k}(Z_{n+1}) - \hat{f}_{i_{n+1},k}(Z_n) \right| \geq c/2$. Using (2), we have either $\phi_0(\hat{f}_{i_{n+1},k}(Z_{n+1}), \delta_{i_{n+1},k}) \geq a$ or $b\mathbf{K}_{i_{n+1},i_{n+1}}/(2\lambda n) \geq c/2$, implying that $\frac{1}{a}\phi_0(\hat{f}_{i_{n+1},k}(Z_{n+1}), \delta_{i_{n+1},k}) + \left( \frac{b\mathbf{K}_{i_{n+1},i_{n+1}}}{c\lambda n} \right)^p \geq 1 = \mathbf{err}(\hat{f}_{i_{n+1},\cdot}(Z_n), y_{i_{n+1}})$. This proves (3).

We are now ready to prove Theorem 1 using (3). For every $j \in Z_{n+1}$, denote by $Z_{n+1}^{(j)}$ the subset of $n$ samples in $Z_{n+1}$ with the $j$-th data point left out. We have $\mathbf{err}(\hat{f}_{j,\cdot}(Z_n^{(j)}), y_j) \leq \frac{1}{a}\phi(\hat{f}_{j,\cdot}(Z_{n+1}), y_j) + \left( \frac{b}{c\lambda n}\mathbf{K}_{j,j} \right)^p$. We thus obtain for all $f \in R^{mK}$:

$$\mathbf{E}_{Z_n} \frac{1}{m-n} \sum_{j \in \bar{Z}_n} \mathbf{err}(\hat{f}_{j,\cdot}(Z_n), y_j) \leq \frac{1}{n+1}\mathbf{E}_{Z_{n+1}} \sum_{j \in Z_{n+1}} \mathbf{err}(\hat{f}_{j,\cdot}(Z_n^{(j)}), y_j)$$

$$\leq \frac{1}{n+1}\mathbf{E}_{Z_{n+1}} \left[ \frac{1}{a} \sum_{j \in Z_{n+1}} \phi(\hat{f}_{j,\cdot}(Z_{n+1}), y_j) + \sum_{j \in Z_{n+1}} \left( \frac{b}{c\lambda n}\mathbf{K}_{j,j} \right)^p \right]$$

$$\leq \frac{n}{a(n+1)}\mathbf{E}_{Z_{n+1}} \left[ \frac{1}{n} \sum_{j \in Z_{n+1}} \phi(f_{j,\cdot}, y_j) + \lambda f^T \mathbf{Q_K} f \right] + \frac{1}{n+1}\mathbf{E}_{Z_{n+1}} \sum_{j \in Z_{n+1}} \left( \frac{b}{c\lambda n}\mathbf{K}_{j,j} \right)^p. \quad \square$$

The formulation used here corresponds to the one-versus-all method for multi-category classification. For the SVM loss $\phi_0(x,y) = \max(0, 1 - (2x-1)(2y-1))$, we may take $a = 0.5$, $b = 2$, and $c = 0.5$. In the experiments reported here, we shall employ the least squares function $\phi_0(x,y) = (x-y)^2$ which is widely used for graph learning. With this formulation, we may choose $a = 1/16$, $b = 0.5$, $c = 0.5$ in Theorem 1.

## 3 Laplacian regularization

Consider an undirected graph $G = (V, E)$ defined on the nodes $V = \{v_j : j = 1, \ldots, m\}$, with edges $E \subset \{1, \ldots, m\} \times \{1, \ldots, m\}$, and weights $w_{j,j'} \geq 0$ associated with edges $(j, j') \in E$. For simplicity, we assume that $(j, j) \notin E$ and $w_{j,j'} = 0$ when $(j, j') \notin E$. Let $\deg_j(G) = \sum_{j'=1}^m w_{j,j'}$ be the degree of node $j$ of graph $G$. We consider the following definition of normalized Laplacian.

**Definition 1** *Consider a graph $G = (V, E)$ of $m$ nodes with weights $w_{j,j'}$ $(j, j' = 1, \ldots, m)$. The unnormalized Laplacian matrix $\mathcal{L}(G) \in R^{m \times m}$ is defined as: $\mathcal{L}_{j,j'}(G) = -w_{j,j'}$ if $j \neq j'$; $\deg_j(G)$ otherwise. Given $m$ scaling factors $\mathbf{S}_j$ $(j = 1, \ldots, m)$, let $\mathbf{S} = \mathrm{diag}(\{\mathbf{S}_j\})$. The $\mathbf{S}$-normalized Laplacian matrix is defined as: $\mathcal{L}_{\mathbf{S}}(G) = \mathbf{S}^{-1/2}\mathcal{L}(G)\mathbf{S}^{-1/2}$. The corresponding regularization is based on: $f_{\cdot,k}^T \mathcal{L}_{\mathbf{S}}(G) f_{\cdot,k} = \frac{1}{2} \sum_{j,j'=1}^m w_{j,j'} \left( \frac{f_{j,k}}{\sqrt{\mathbf{S}_j}} - \frac{f_{j',k}}{\sqrt{\mathbf{S}_{j'}}} \right)^2$.*

A common choice of $\mathbf{S}$ is $\mathbf{S} = \mathbf{I}$, corresponding to regularizing with the unnormalized Laplacian $\mathcal{L}$. The idea is natural: we assume that the predictive values $f_{j,k}$ and $f_{j',k}$ should be close when $(j, j') \in E$ with a strong link. Another common choice is to normalize by $\mathbf{S}_j = \deg_j(G)$ (i.e. $\mathbf{S} = \mathbf{D}$) so that diagonals of $\mathcal{L}_{\mathbf{S}}$ become all one [3, 4, 7, 2].

**Definition 2** *Given label $y = \{y_j\}_{j=1,\ldots,m}$ on $V$, we define the cut for $\mathcal{L}_{\mathbf{S}}$ in Definition 1 as: $\mathbf{cut}(\mathcal{L}_{\mathbf{S}}, y) = \sum_{j,j':y_j \neq y_{j'}} \frac{w_{j,j'}}{2} \left( \frac{1}{\mathbf{S}_j} + \frac{1}{\mathbf{S}_{j'}} \right) + \sum_{j,j':y_j = y_{j'}} \frac{w_{j,j'}}{2} \left( \frac{1}{\sqrt{\mathbf{S}_j}} - \frac{1}{\sqrt{\mathbf{S}_{j'}}} \right)^2$.*

Unlike typical graph-theoretical definitions of graph-cut, this learning theoretical definition of graph-cut penalizes not only between-class edge weights but also within-class edge weights when such an edge connects two nodes with different scaling factors. This penalization is intuitive if we look at the regularizer in Definition (1), which encourages $f_{j,k}/\sqrt{\mathbf{S}_j}$ to be similar to $f_{j',k}/\sqrt{\mathbf{S}_{j'}}$ when $w_{j,j'}$ is large. If $j$ and $j'$ belongs to the same class, we want $f_{j,k}$ to be similar to $f_{j',k}$. Therefore for such

an in-class pair $(j, j')$, we want to have $\mathbf{S}_j \approx \mathbf{S}_{j'}$. This penalization has important consequences, which we will investigate later in the paper. For unnormalized Laplacian (i.e. $\mathbf{S}_j = 1$), the second term on the right hand side of Definition 2 vanishes, and our learning theoretical definition becomes identical to the standard graph-theoretical definition: $\mathbf{cut}(\mathcal{L}, y) = \sum_{j,j':y_j \neq y_{j'}} w_{j,j'}$.

We consider $\mathbf{K}$ in (1) defined as follows: $\mathbf{K} = (\alpha \mathbf{S}^{-1} + \mathcal{L}_{\mathbf{S}}(G))^{-1}$, where $\alpha > 0$ is a tuning parameter to make $\mathbf{K}$ strictly positive definite. This parameter is important.

For simplicity, we state the generalization bound based on Theorem 1 with optimal $\lambda$. Note that in applications, $\lambda$ is usually tuned through cross validation. Therefore assuming optimal $\lambda$ will simplify the bound so that we can focus on the more essential characteristics of generalization performance.

**Theorem 2** *Let the conditions in Theorem 1 hold with the regularization condition $\mathbf{K} = (\alpha \mathbf{S}^{-1} + \mathcal{L}_{\mathbf{S}}(G))^{-1}$. Assume that $\phi_0(0,0) = \phi_0(1,1) = 0$, then $\forall p > 0$, there exists a sample independent regularization parameter $\lambda$ in (1) such that the expected generalization error is bounded by:*

$$\mathbf{E}_{Z_n} \frac{1}{m-n} \sum_{j \in \bar{Z}_n} \mathbf{err}(\hat{f}_{j,\cdot}(Z_n), y_j) \leq \frac{C_p(a,b,c)}{n^{p/(p+1)}}(\alpha s + \mathbf{cut}(\mathcal{L}_{\mathbf{S}}, y))^{p/(p+1)} \mathbf{tr}_p(\mathbf{K})^{p/(p+1)},$$

*where $C_p(a,b,c) = (b/ac)^{p/(p+1)}(p^{1/(p+1)} + p^{-p/(p+1)})$ and $s = \sum_{j=1}^m \mathbf{S}_j^{-1}$.*

*Proof.* Let $f_{j,k} = \delta_{y_j,k}$. It can be easily verified that $\sum_{j=1}^m \phi(f_{j,\cdot}, y_j)/m + \lambda f^T \mathbf{Q}_{\mathbf{K}} f = \lambda(\alpha s + \mathbf{cut}(\mathcal{L}_{\mathbf{S}}, y))$. Now, we simply use this expression in Theorem 1, and then optimize over $\lambda$. $\square$

This theorem relates graph-cut to generalization performance. The conditions on the loss function in Theorem 2 hold for least squares with $b/ac = 16$. It also applies to other standard loss functions such as SVM. With $p$ fixed, the generalization error decreases at the rate $O(n^{-p/(p+1)})$ when $n$ increases. This rate of convergence is faster when $p$ increases. However in general, $\mathbf{tr}_p(\mathbf{K})$ is an increasing function of $p$. Therefore we have a trade-off between the two terms. The bound also suggests that if we normalize the diagonal entries of $\mathbf{K}$ such that $\mathbf{K}_{j,j}$ is a constant, then $\mathbf{tr}_p(\mathbf{K})$ is independent of $p$, and thus a larger $p$ can be used in the bound. This motivates the idea of normalizing the diagonals of $\mathbf{K}$. Our goal is to better understand how the quantity $(\alpha s + \mathbf{cut}(\mathcal{L}_{\mathbf{S}}, y))^{\frac{p}{p+1}} \mathbf{tr}_p(\mathbf{K})^{\frac{p}{p+1}}$ is related to properties of the graph, which gives better understanding of graph-based learning.

**Definition 3** *A subgraph $G_0 = (V_0, E_0)$ of $G = (V, E)$ is a pure component if $G_0$ is connected, $E_0$ is induced by restricting $E$ on $V_0$, and if labels $y$ have identical values on $V_0$. A pure subgraph $G' = \cup_{\ell=1}^q G_\ell$ of $G$ divides $V$ into $q$ disjoint sets $V = \cup_{\ell=1}^q V_\ell$ such that each subgraph $G_\ell = (V_\ell, E_\ell)$ is a pure component. Denote by $\lambda_i(G_\ell) = \lambda_i(\mathcal{L}(G_\ell))$ the $i$-th smallest eigenvalue of $\mathcal{L}(G_\ell)$.*

If we remove all edges of $G$ that connect nodes with different labels, then the resulting subgraph is a pure subgraph (but not the only one). For each pure component $G_\ell$, its first eigenvalue $\lambda_1(G_\ell)$ is always zero. The second eigenvalue $\lambda_2(G_\ell) > 0$, and it measures how well-connected $G_i$ is [2].

**Theorem 3** *Let the assumptions of Theorem 2 hold, and $G' = \cup_{\ell=1}^q G_\ell$ $(G_\ell = (V_\ell, E_\ell))$ be a pure subgraph of $G$. For all $p \geq 1$, there exist sample-independent $\lambda$ and $\alpha$, such that the generalization performance of (1), $\mathbf{E}_{Z_n} \sum_{j \in \bar{Z}_n} \mathbf{err}(\hat{f}_{j,\cdot}, y_j)/(m-n)$, is bounded by*

$$\frac{C_p(a,b,c)}{n^{p/(p+1)}} \left( s^{1/2} \left( \sum_{\ell=1}^q \frac{s_\ell(p)/m}{m_\ell^p} \right)^{1/2p} + \mathbf{cut}(\mathcal{L}_{\mathbf{S}}, y)^{1/2} \left( \sum_{\ell=1}^q \frac{s_\ell(p)/m}{\lambda_2(G_\ell)^p} \right)^{1/2p} \right)^{2p/(p+1)},$$

*where $m_\ell = |V_\ell|$, $s = \sum_{j=1}^m \mathbf{S}_j^{-1}$, and $s_\ell(p) = \sum_{j \in V_\ell} \mathbf{S}_j^p$.*

*Proof sketch.* We simply upper bound $\mathbf{tr}_p(\mathbf{K})$ in terms of $\lambda_2(G_\ell)$ and $s_\ell$, where $\mathbf{K} = (\alpha \mathbf{S}^{-1} + \mathcal{L}_{\mathbf{S}})^{-1}$. Substitute this estimation into Theorem 2 and optimize it over $\alpha$. $\square$

To put this into perspective, suppose that we use unnormalized Laplacian regularizer on a zero-cut graph. Then $\mathbf{S} = \mathbf{I}$ and $\mathbf{cut}(\mathcal{L}_{\mathbf{S}}, y) = 0$, and by letting $p = 1$ and $p \to \infty$ in Theorem 3, we have:

$$\mathbf{E}_{Z_n} \sum_{j \in \bar{Z}_n} \frac{\mathbf{err}(\hat{f}_{j,\cdot}, y_j)}{m-n} \leq 2\sqrt{\frac{b}{ac} \cdot \frac{q}{n}} \quad \text{and} \quad \mathbf{E}_{Z_n} \sum_{j \in \bar{Z}_n} \frac{\mathbf{err}(\hat{f}_{j,\cdot}, y_j)}{m-n} \leq \frac{b}{ac} \cdot \frac{m}{n \min_\ell m_\ell}.$$

That is, in the zero-cut case, the generalization performance can be bounded as $O(\sqrt{q/n})$. We can also achieve a faster convergence rate of $O(1/n)$, but it also depends on $m/(\min_\ell m_\ell) \geq q$. This implies that we will achieve better convergence at the $O(1/n)$ level if the sizes of the components are balanced, while the convergence may behave like $O(\sqrt{q/n})$ otherwise.

### 3.1 Near zero-cut optimum scaling factors

The above observation motivates a scaling matrix $\mathbf{S}$ so that it compensates for the unbalanced pure component sizes. From Definition 2 and Theorem 2 we know that good scaling factors should be approximately constant within each class. Here we focus on the case that scaling factors are constant within each pure component ($\mathbf{S}_j = \bar{s}_\ell$ when $j \in V_\ell$) in order to derive optimum scaling factors.

Let us define $\mathbf{cut}(G', y) = \sum_{j,j':y_j \neq y_{j'}} w_{j,j'} + \sum_{\ell \neq \ell'} \sum_{j \in V_\ell, j' \in V_{\ell'}} \frac{w_{j,j'}}{2}$. In Theorem 3, when we use $\mathbf{cut}(\mathcal{L}_\mathbf{S}, y) \leq \mathbf{cut}(G', y)/\min_\ell \bar{s}_\ell$ and let $p \to \infty$ and assume that $\mathbf{cut}(G', y)$ is sufficiently small, the dominate term of the bound becomes $\frac{\max_\ell(\bar{s}_\ell/m_\ell)}{n} \sum_{\ell=1}^q \frac{m_\ell}{\bar{s}_\ell}$, which can then be optimized with the choice $\bar{s}_\ell = m_\ell$, and the resulting bound becomes:

$$\frac{1}{m-n} \sum_{j \in \bar{Z}_n} \mathbf{err}(\hat{f}_{j,\cdot}, y_j) \leq \frac{b}{ac} \cdot \frac{1}{n} \left( \sqrt{q} + \sqrt{\frac{\mathbf{cut}(G', y)}{u(G') \min_\ell m_\ell}} \right)^2,$$

where $u(G') = \min_\ell(\lambda_2(G_\ell)/m_\ell)$. Hence, if $\mathbf{cut}(G', y)$ is small, then we should choose $\bar{s}_\ell \propto m_\ell$ for each pure component $\ell$ so that the generalization performance is approximately $(ac)^{-1}b \cdot q/n$.

The analysis provided here not only formally shows the importance of normalization in the learning theoretical framework but also suggests that the good normalization factor for each node $j$ is approximately the size of the well-connected pure component that contains node $j$ (assuming that nodes belonging to different pure components are only weakly connected). The commonly practiced degree-based normalization method $\mathbf{S}_j = \deg_j(G)$ provides such good normalization factors under a simplified "box model" used in early studies e.g. [4]. In this model, each node connects to itself and all other nodes of the same pure component with edge weight $w_{j,j'} = 1$. The degree is thus $\deg_j(G_\ell) = |V_\ell| = m_\ell$, which gives the optimal scaling in our analysis. However, in general, the box model may not be a good approximation for practical problems. A more realistic approximation, which we call core-satellite model, will be introduced in the experimental section. For such a model, the degree-based normalization can fail because the $\deg_j(G_\ell)$ within each pure component $G_\ell$ is not approximately constant (thus raising $\mathbf{cut}(\mathcal{L}_\mathbf{S}, y)$), and it may not be proportional to $m_\ell$.

Our remedy is as follows. Let $\bar{\mathbf{K}} = (\alpha \mathbf{I} + \mathcal{L})^{-1}$ be the kernel matrix corresponding to the unnormalized Laplacian. Let $v_\ell \in R^m$ be the vector whose $j$-th entry is 1 if $j \in V_\ell$ and 0 otherwise. Then it is easy to verify that for small $\alpha$ and near-zero $\mathbf{cut}(G', y)$, we have $\alpha \bar{\mathbf{K}} = \sum_{\ell=1}^q v_\ell v_\ell^T / m_\ell + O(1)$, and thus $\bar{\mathbf{K}}_{j,j} \propto m_\ell^{-1}$ for each $j \in V_\ell$. Therefore the scaling factor $\mathbf{S}_j = 1/\bar{\mathbf{K}}_{j,j}$ is nearly optimal for all $j$. We call this method of normalization ($\mathbf{S}_j = 1/\bar{\mathbf{K}}_{j,j}$, $\mathbf{K} = (\alpha \mathbf{S}^{-1} + \mathcal{L}_\mathbf{S})^{-1}$) $\mathbf{K}$-*scaling* in this paper as it scales the kernel matrix $\mathbf{K}$ so that each $\mathbf{K}_{j,j} = 1$. By contrast, we call the standard degree-based normalization ($\mathbf{S}_j = \deg_j(G)$, $\mathbf{K} = (\alpha \mathbf{I} + \mathcal{L}_\mathbf{S})^{-1}$) $\mathbf{L}$-*scaling* as it scales diagonals of $\mathcal{L}_\mathbf{S}$ to 1. Although $\mathbf{K}$-scaling coincides with a common practice in standard kernel learning, it is important to notice that showing this method behaves well in the graph learning setting is non-trivial and novel. In fact, no one has proposed this normalization method in the graph learning setting before this work. Without the learning theoretical results developed here, it is not obvious whether this method should work better than the commonly practiced degree-based normalization.

## 4 Dimension Reduction

Normalization and dimension reduction have been commonly used in spectral clustering such as [3, 4]. For semi-supervised learning, dimension reduction (without normalization) is known to improve performance [1, 6] while normalization (without dimension reduction) has also been explored [7]. An appropriate combination of normalization and dimension reduction can further improve performance. We shall first introduce dimension reduction with normalized Laplacian $\mathcal{L}_\mathbf{S}(G)$. Denote by $\mathbf{P}_\mathbf{S}^r(G)$ the projection operator onto the eigenspace of $\alpha \mathbf{S}^{-1} + \mathcal{L}_\mathbf{S}(G)$ corresponding to the $r$

smallest eigenvalues. Now, we may define the following regularizer on the reduced subspace:

$$f_{\cdot,k}^T \mathbf{K}^{-1} f_{\cdot,k} = \begin{cases} f_{\cdot,k}^T \mathbf{K}_0^{-1} f_{\cdot,k} & \mathbf{P}_{\mathbf{S}}^r(G) f_{\cdot,k} = f_{\cdot,k}, \\ +\infty & \text{otherwise.} \end{cases} \tag{4}$$

Note that we will focus on bounding the generalization complexity using the reduced dimensionality $r$. In such context, the choice of $\mathbf{K}_0$ is not important. For example, we may simply choose $\mathbf{K}_0 = \mathbf{I}$. The benefit of dimension reduction in graph learning has been investigated in [6], under the spectral kernel design framework. Note that the normalization issue, which will change the eigenvectors and their ordering, wasn't investigated there. The following theorem shows that the target vectors can be well approximated by its projection onto $\mathbf{P}_{\mathbf{S}}^q(G)$. We skip the proof due to the space limitation.

**Theorem 4** *Let $G' = \cup_{\ell=1}^q G_\ell$ ($G_\ell = (V_\ell, E_\ell)$) be a pure subgraph of $G$. Consider $r \geq q$: $\lambda_{r+1}(\mathcal{L}_{\mathbf{S}}(G)) \geq \lambda_{r+1}(\mathcal{L}_{\mathbf{S}}(G')) \geq \min_\ell \lambda_2(\mathcal{L}_{\mathbf{S}}(G_\ell))$. For each $k$, let $\bar{f}_{j,k} = \delta_{y_j,k}$ be the target (encoding of the true labels) for class $k$ ($j = 1, \ldots, m$). Then $\|\mathbf{P}_{\mathbf{S}}^r(G)\bar{f}_{\cdot,k} - \bar{f}_{\cdot,k}\|_2^2 \leq \delta_r(\mathbf{S})\|\bar{f}_{\cdot,k}\|_2^2$, where $\delta_r(\mathbf{S}) = \frac{\|\mathcal{L}_{\mathbf{S}}(G) - \mathcal{L}_{\mathbf{S}}(G')\|_2 + d(\mathbf{S})}{\lambda_{r+1}(\mathcal{L}_{\mathbf{S}}(G))}$, $d(\mathbf{S}) = \max_\ell \frac{1}{2|V_\ell|} \sum_{j,j' \in V_\ell} (\mathbf{S}_j^{-1/2} - \mathbf{S}_{j'}^{-1/2})^2$.*

We can prove a generalization bound using Theorem 4. For simplicity, we only consider least squares loss $\phi(f_{j,\cdot}, y_j) = \sum_{k=1}^K (f_{j,k} - \delta_{k,y_j})^2$ in (1) using regularization (4) and $\mathbf{K}_0 = \mathbf{I}$. With $p = 1$, we have $\frac{1}{m} \sum_{j=1}^m \phi(\bar{f}_{j,\cdot}, y_j) \leq \delta_r(\mathbf{S})^2 + \lambda m$. It is also equivalent to take $\mathbf{K}_0 = \mathbf{P}_{\mathbf{S}}^r(G)$ due to the dimension reduction, so that we can use $\mathbf{tr}(\mathbf{K}) = r$. Now from Theorem 1 with $a = 1/16$, $b = 0.5, c = 0.5$, we have $\mathbf{E}_{Z_n} \frac{1}{m-n} \sum_{j \in \bar{Z}_n} \mathbf{err}(\hat{f}_{j,\cdot}, y_j) \leq 16(\delta_r(\mathbf{S})^2 + \lambda m) + \frac{r}{\lambda nm}$. By optimizing over $\lambda$, we obtain

$$\mathbf{E}_{Z_n} \sum_{j \in \bar{Z}_n} \frac{\mathbf{err}(\hat{f}_{j,\cdot}, y_j)}{m-n} \leq 16\delta_r(\mathbf{S})^2 + 32\sqrt{r/n}. \tag{5}$$

The analysis of optimum scaling factors is analogous to Section 3.1, and the conclusions there hold. Compared to Theorem 3, the advantage of dimension reduction in (5) is that the quantity $\mathbf{cut}(\mathcal{L}_{\mathbf{S}}, y)$ is replaced by $\|\mathcal{L}_{\mathbf{S}}(G) - \mathcal{L}_{\mathbf{S}}(G')\|_2$, which is typically much smaller. Instead of a rigorous analysis, we shall just give a brief intuition. For simplicity we take $\mathbf{S} = \mathbf{I}$ so that we can ignore the variations caused by $\mathbf{S}$. The 2-norm of the symmetric error matrix $\mathcal{L}_{\mathbf{S}}(G) - \mathcal{L}_{\mathbf{S}}(G')$ is its largest eigenvalue, which is no more than the largest 1-norm of one of its row vectors. In contrast, $\mathbf{cut}(\mathcal{L}_{\mathbf{S}}, y)$ behaves similar to the absolute sum of entries of the error matrix, which is $m$ times more than the averaged 1-norm of its row vectors. Therefore if error is relatively uniform across rows, then $\mathbf{cut}(\mathcal{L}_{\mathbf{S}}, y)$ can be at an order of $m$ times more than $\|\mathcal{L}_{\mathbf{S}}(G) - \mathcal{L}_{\mathbf{S}}(G')\|_2$.

## 5 Experiments

We test the three types of the kernel matrix $\mathbf{K}$ (*Unnormalized*, normalized by $\mathbf{K}$-*scaling* or $\mathbf{L}$-*scaling*) with the two regularization methods: the first method is to use $\mathbf{K}$ without dimension reduction, and the second method reduces the dimension of $\mathbf{K}^{-1}$ to eigenvectors corresponding to the smallest $r$ eigenvalues and regularizes with $f^T \mathbf{K}^{-1} f$ if $\mathbf{P}_{\mathbf{S}}^r(G) f = f$ and $+\infty$ otherwise. We are particularly interested in how well $\mathbf{K}$-scaling performs. From $m$ data points, $n$ training labeled examples are randomly chosen while ensuring that at least one training example is chosen from each class. The remaining $m - n$ data points serve as test data. The regularization parameter $\lambda$ is chosen by cross validation on the $n$ training labeled examples. We will show performance either when the rest of the parameters ($\alpha$ and dimensionality $r$) are also chosen by cross validation or when they are set to the optimum (*oracle* performance). The dimensionality $r$ is chosen from $K, K+5, K+10, \cdots, 100$ where $K$ is the number of classes unless otherwise specified. Our focus is on small $n$ close to the number of classes. Throughout this section, we conduct 10 runs with random training/test splits and report the average accuracy. We use the one-versus-all strategy with least squares loss $\phi_k(a, b) = (a - \delta_{k,b})^2$.

**Controlled data experiments**

The purpose of the controlled data experiments is to observe the correlation of the effectiveness of the normalization methods with graph properties. The graphs we generate contain 2000 nodes, each of which is assigned one of 10 classes. We show the results when dimension reduction is applied

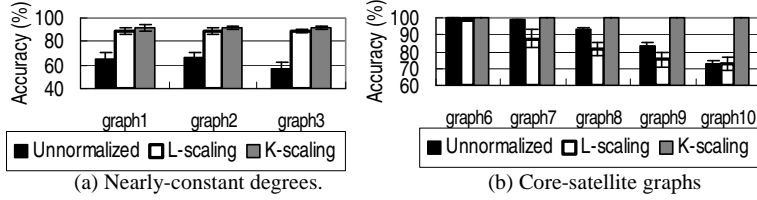

(a) Nearly-constant degrees.　　　(b) Core-satellite graphs

Figure 1: Classification accuracy (%). (a) Graphs with near constant within class degrees. (b) Core-satellite graphs. $n = 40, m = 2000$. With dimension reduction (dim $\leq 20$; chosen by cross validation).

to the three types of matrix $\mathbf{K}$. The performance is averaged over 10 random splits with error bar representing one standard deviation. Figure 1 (a) shows classification accuracy on three graphs that were generated so that the node degrees (of either correct edges or erroneous edges) are close to constant within each class but vary across classes. On these graphs, both $\mathbf{K}$-scaling and $\mathbf{L}$-scaling significantly improve classification accuracy over the unnormalized baseline. There is not much difference between $\mathbf{K}$-scaling's and $\mathbf{L}$-scaling's. Observe that $\mathbf{K}$-scaling and $\mathbf{L}$-scaling perform differently on the graphs used in Figure 1 (b). These five graphs have the following properties. Each class consists of *core nodes* and *satellite nodes*. Core nodes of the same class are tightly connected with each other and do not have any erroneous edges. Satellite nodes are relatively weakly connected to core nodes of the same class. They are also connected to some other classes' satellite nodes (i.e., introducing errors). This core-satellite model is intended to simulate real-world data in which some data points are close to the class boundaries (satellite nodes). For graphs generated in this manner, degrees vary within the same class since the satellite nodes have smaller degrees than the core nodes. Our analysis suggests that $\mathbf{L}$-scaling will do poorly. Figure 1 (b) shows that on the five core-satellite graphs, $\mathbf{K}$-scaling indeed produces higher performance than $\mathbf{L}$-scaling. In particular, $\mathbf{K}$-scaling does well even when $\mathbf{L}$-scaling rather underperforms the unnormalized baseline.

**Real-world data experiments**

Our real-world data experiments use an image data set (MNIST) and a text data set (RCV1). The MNIST data set, downloadable from http://yann.lecun.com/exdb/mnist/, consists of hand-written digit image data (representing 10 classes, from digit "0" to "9"). For our experiments, we randomly choose 2000 images (i.e., $m = 2000$). Reuters Corpus Version 1 (RCV1) consists of news articles labeled with topics. For our experiments, we chose 10 topics (ranging from sports to labor issues; representing 10 classes) that have relatively large populations and randomly chose 2000 articles that are labeled with exactly one of those 10 topics. To generate graphs from the image data, as is commonly done, we first generate the vectors of the gray-scale values of the pixels, and produce the edge weight between the $i$-th and the $j$-th data points $\mathbf{X}_i$ and $\mathbf{X}_j$ by $w_{i,j} = \exp(-||\mathbf{X}_i - \mathbf{X}_j||^2/t)$ where $t > 0$ is a parameter (RBF kernels). To generate graphs from the text data, we first create the bag-of-word vectors and then set $w_{i,j}$ based on RBF as above. As our baseline, we test the supervised configuration by letting $\mathbf{W} + \beta\mathbf{I}$ be the kernel matrix and using the same least squares loss function, where we use the *oracle* $\beta$ which is optimal.

Figures 2 (a-1,2) shows performance in relation to the number of labeled examples ($n$) on the MNIST data set. The comparison of the three bold lines (representing the methods with dimension reduction) in Figure 2 (a-1) shows that when the dimensionality and $\alpha$ are determined by cross validation,

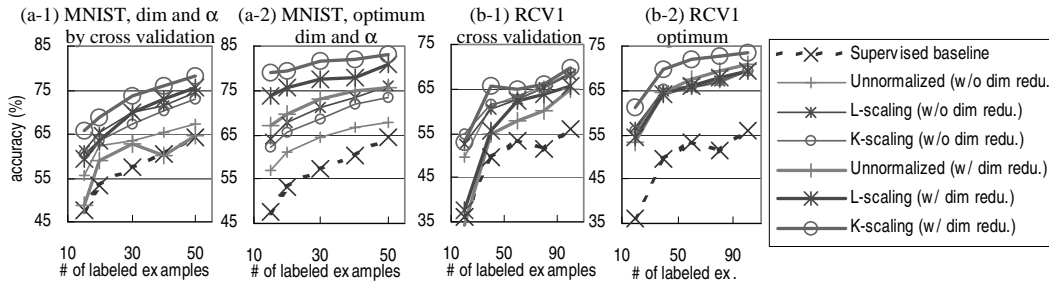

Figure 2: Classification accuracy (%) versus sample size $n$ ($m = 2000$). (a-1) MNIST, dim and $\alpha$ determined by cross validation. (a-2) MNIST, dim and $\alpha$ set to the optimum. (b-1) RCV1, dim and $\alpha$ determined by cross validation. (b-2) RCV1, dim and $\alpha$ set to the optimum.

**K**-scaling outperforms **L**-scaling, and **L**-scaling outperforms the unnormalized Laplacian. The performance differences among these three are statistically significant ($p \leq 0.01$) based on the paired $t$ test. The performance of the unnormalized Laplacian (with dimension reduction) is roughly consistent with the performance with similar $(m, n)$ with heuristic dimension selection in [1]. Without dimension reduction, **L**-scaling and **K**-scaling still improve performance over the unnormalized Laplacian. The best performance is always obtained by **K**-scaling with dimension reduction.

In Figure 2 (a-1), the unnormalized Laplacian with dimension reduction underperforms the unnormalized Laplacian without dimension reduction, indicating that dimension reduction rather degrades performance. By comparing Figure 2 (a-1) and (a-2), we observe that this seemingly counter-intuitive performance trend is caused by the difficulty of choosing the right dimensionality by cross validation. Figure 2 (a-2) shows the performance at the *oracle* optimal dimensionality and $\alpha$. As observed, if the optimal dimensionality is known (as in (a-2)), dimension reduction improves performance either with or without normalization by **K**-scaling and **L**-scaling, and all transductive configurations outperform the supervised baseline. We also note that the comparison of Figure 2 (a-1) and (a-2) shows that choosing good dimensionality by cross validation is much harder than choosing $\alpha$ by cross validation, especially when the number of labeled examples is small. On the RCV1 data set, the performance trend is similar to that of MNIST. Figures 2 (b-1,2) shows the performance on RCV1 using the RBF kernel ($t = 0.25$, 100NN). In the setting of Figure 2 (b-1) where the dimensionality and $\alpha$ were determined by cross validation, **K**-scaling with dimension reduction generally performs the best. By setting the dimensionality and $\alpha$ to the optimum, the benefit of **K**-scaling with dimension reduction is even clearer (Figure 2 (b-2)). Its performance differences from the second and third best '**L**-scaling (w/ dim redu.)' and 'Unnormalized (w/ dim redu.)' are statistically significant ($p \leq 0.01$) in both Figure 2 (b-1) and (b-2).

In our experiments, **K**-scaling with dimension reduction consistently outperformed others. Without dimension reduction, **K**-scaling and **L**-scaling are not always effective. This is consistent with our analysis. On real data, **cut** is not near-zero, and the effect of normalization is unclear (Section 3.1); however, when dimension is reduced, $\|\mathcal{L}_{\mathbf{S}}(G) - \mathcal{L}_{\mathbf{S}}(G')\|_2$ (corresponding to **cut**) can be much smaller (Section 4), which suggests that **K**-scaling should improve performance.

## 6 Conclusion

We derived generalization bounds for learning on graphs with Laplacian regularization, using properties of the graph. In particular, we explained the importance of Laplacian normalization and dimension reduction for graph learning. We argued that the standard **L**-scaling normalization method has the undesirable property that the normalization factors can vary significantly within a pure component. An alternate normalization method, which we call **K**-scaling, is proposed to remedy the problem. Experiments confirm the superiority of the this normalization scheme.

## References

[1] M. Belkin and P. Niyogi. Semi-supervised learning on Riemannian manifolds. *Machine Learning*, Special Issue on Clustering:209–239, 2004.

[2] F. R. Chung. *Spectral Graph Theory*. Regional Conference Series in Mathematics. American Mathematical Society, Rhode Island, 1998.

[3] A. Y. Ng, M. I. Jordan, and Y. Weiss. On spectral clustering: Analysis and an algorithm. In *NIPS*, pages 849–856, 2001.

[4] J. Shi and J. Malik. Normalized cuts and image segmentation. *IEEE Trans. Pattern Anal. Mach. Intell*, 22:888–905, 2000.

[5] M. Szummer and T. Jaakkola. Partially labeled classification with Markov random walks. In *NIPS 2001*, 2002.

[6] T. Zhang and R. K. Ando. Analysis of spectral kernel design based semi-supervised learning. In *NIPS*, 2006.

[7] D. Zhou, O. Bousquet, T. Lal, J. Weston, and B. Schlkopf. Learning with local and global consistency. In *NIPS 2003*, pages 321–328, 2004.

[8] X. Zhu, Z. Ghahramani, and J. Lafferty. Semi-supervised learning using Gaussian fields and harmonic functions. In *ICML 2003*, 2003.
